# Speeding up Magnetic Resonance Image Acquisition by Bayesian Multi-Slice Adaptive Compressed Sensing

**Matthias W. Seeger**
Saarland University and Max Planck Institute for Informatics
Campus E1.4, 66123 Saarbrücken, Germany
mseeger@mmci.uni-saarland.de

## Abstract

We show how to sequentially optimize magnetic resonance imaging measurement designs over stacks of neighbouring image slices, by performing convex variational inference on a large scale non-Gaussian linear dynamical system, tracking dominating directions of posterior covariance without imposing any factorization constraints. Our approach can be scaled up to high-resolution images by reductions to numerical mathematics primitives and parallelization on several levels. In a first study, designs are found that improve significantly on others chosen independently for each slice or drawn at random.

## 1 Introduction

Magnetic resonance imaging (MRI) [10, 6] is a very flexible imaging modality. Inflicting no harm on patients, it is used for an ever-growing number of diagnoses in health-care. Its most serious limitation is acquisition speed, being based on a serial idea (gradient encoding) with limited scope for parallelization. Fourier (aka. $k$-space) coefficients are sampled along smooth trajectories (phase encodes), many of which are needed for reconstructions of sufficient quality [17, 1]. Long scan times lead to patient annoyance, grave errors due to movement, and high running costs. The Nyquist sampling theorem [2] fundamentally limits traditional linear image reconstruction, but with modern 3D MRI scenarios, dense sampling is not practical anymore. Acquisition is accelerated to some extent in parallel MRI[1], by using receive coil arrays [19, 9]: the sensitivity profiles of different coils provide part of the localization normally done by more phase steps. A different idea is to use (nonlinear) sparse image reconstruction, with which the Nyquist limit can be undercut robustly for images, emphasized recently as compressed sensing [5, 3]. While sparse reconstruction has been used for MRI [28, 12], we address the more fundamental question of how to *optimize* the sampling design for sparse reconstruction over a specific real-world signal class (MR images) in an adaptive manner, avoiding strong assumptions such as exact, randomly distributed sparsity that do not hold for real images [23]. Our approach is in line with recent endeavours to extend MRI capabilities and reduce its cost, by complementing expensive, serial hardware with easily parallelizable digital computations.

We extend the framework of [24], the first approximate Bayesian method for MRI sampling optimization applicable at resolutions of clinical interest. Their approach falls short of real MRI practice on a number of points. They considered single image slices only, while stacks[2] of neighbouring

slices are typically acquired. Reconstruction can be improved significantly by taking the strong statistical dependence between pixels of nearby slices into account [14, 26, 18]. Design optimization is a joint problem as well: using the same acquisition pattern for neighbouring slices is clearly redundant. Second, the latent image was modelled as real-valued in [24], while in reality it is a complex-valued signal. To our knowledge, the few directly comparable approaches rely on "trial-and-error" exploration [12, 16, 27], requiring substantially more human expert interventions and real MRI measurements, whose high costs our goal-directed method aims to minimize.

Our extension to stacks of slices requires new technology. Global Gaussian covariances have to be approximated, a straightforward extension of which to many slices is out of the question. We show how to use approximate Kalman smoothing, implementing message passing by the Lanczos algorithm, which has not been done in machine learning before (see [20, 25] for similar proposals to oceanography problems). Our technique is complementary to mean field variational inference approximations ("variational Bayes"), where most correlations are ruled out *a priori*. We track the dominating posterior covariance directions inside our method, allowing them to change during optimization. While our double loop approach may be technically more demanding to implement, relaxation as well as algorithm are characterized much better (convex problem; algorithm reducing to standard computational primitives), running orders of magnitude faster. Beyond MRI, applications could be to Bayesian inference over video streams, or to computational photography [11]. Our approach is parallelizable on several levels. This property is essential to even start projecting such applications: on the scale demanded by modern MRI applications, with practitioners being used to view images directly after acquisition, little else but highly parallelizable approaches are viable.

Large scale variational inference is reviewed and extended to complex-valued data in Section 2, lifted to non-Gaussian linear dynamical systems in Section 3, and the experimental design extension is given in Section 4. Results of a preliminary study on data from a Siemens 3T scanner are provided in Section 5, using a serial implementation.

## 2   Large Scale Sparse Inference

Our motivation is to improve MR image reconstruction, not by finding a better estimation technique, but by sampling data more economically. A latent MR image slice $\boldsymbol{u} \in \mathbb{C}^n$ ($n$ pixels) is measured by a design matrix $\boldsymbol{X} \in \mathbb{C}^{m \times n}$: $\boldsymbol{y} = \boldsymbol{X}\boldsymbol{u} + \boldsymbol{\varepsilon}$ ($\boldsymbol{\varepsilon} \sim N(\boldsymbol{0}, \sigma^2 \boldsymbol{I})$ models noise). For Cartesian MRI, $\boldsymbol{X} = \boldsymbol{I}_{S,.} \mathcal{F}_n$, $\mathcal{F}_n$ the 2D fast Fourier transform, $S \subset \{1, \ldots, n\}$ the sampling pattern (which partitions into complete columns or rows: *phase encodes*, the atomic units of the design). Sparse reconstruction works by encoding super-Gaussian image statistics in a non-Gaussian prior, then finding the posterior mode (MAP estimation): a convex quadratic program for the model employed here. To improve the measurement design $\boldsymbol{X}$ itself, posterior information beyond (and independent of) its mode is required, chiefly posterior *covariances*.

We briefly review [24], extending it to complex-valued $\boldsymbol{u}$. The super-Gaussian image prior $P(\boldsymbol{u})$ is adapted by placing potentials on absolute values $|s_j|$, the posterior has the form

$$P(\boldsymbol{u}|\boldsymbol{y}) \propto N(\boldsymbol{y}|\boldsymbol{X}\boldsymbol{u}, \sigma^2 \boldsymbol{I}) \prod_{j=1}^{q} e^{-\tau_j |s_j|/\sigma}, \quad \boldsymbol{s} = \boldsymbol{B}\boldsymbol{u} \in \mathbb{C}^q.$$

Here, $\boldsymbol{B}$ is a sparsity transform [24]. We use the $\mathbb{C} \to \mathbb{R}^2$ embedding, $\boldsymbol{s} = (\boldsymbol{s}_j)$, $\boldsymbol{s}_j \in \mathbb{R}^2$, and norm potentials $e^{-\tau_j \|\boldsymbol{s}_j/\sigma\|}$. Two main ideas lead to [24]. First, inference is relaxed to an optimization problem by lower-bounding the log partition function [7] (intuitively, each Laplace potential $e^{-\tau_j \|\boldsymbol{s}_j/\sigma\|}$ is lower-bounded by a Gaussian-form potential of variance $\gamma_j > 0$), leading to

$$\phi(\boldsymbol{\gamma}) = \log |\boldsymbol{A}| + h(\boldsymbol{\gamma}) + \min_{\boldsymbol{u}} R(\boldsymbol{u}, \boldsymbol{\gamma}), \quad R := \sigma^{-2} \left( \|\boldsymbol{y} - \boldsymbol{X}\boldsymbol{u}\|^2 + \boldsymbol{s}^T \boldsymbol{\Gamma}^{-1} \boldsymbol{s} \right), \; \boldsymbol{\gamma} = (\gamma_j), \; (1)$$

$h(\boldsymbol{\gamma}) = (\boldsymbol{\tau}^2)^T \boldsymbol{\gamma}$. This procedure implies a Gaussian approximation $Q(\boldsymbol{u}|\boldsymbol{y}) = N(\boldsymbol{u}|\boldsymbol{u}_*, \sigma^2 \boldsymbol{A}^{-1})$ to $P(\boldsymbol{u}|\boldsymbol{y})$, with $\boldsymbol{A} = \boldsymbol{X}^H \boldsymbol{X} + \boldsymbol{B}^T \boldsymbol{\Gamma}^{-1} \boldsymbol{B}$ and $\boldsymbol{u}_* = \boldsymbol{u}_*(\boldsymbol{\gamma})$. The complex extension is formally similar to [24] ($\boldsymbol{\pi}$ there is $\boldsymbol{\gamma}^{-1}$ here): $\boldsymbol{\Gamma} := (\mathrm{diag}\,\boldsymbol{\gamma}) \otimes \boldsymbol{I}_2 = \mathrm{diag}(\gamma_1, \gamma_1, \gamma_2, \ldots)^T$, $\boldsymbol{B} := \boldsymbol{B}_{\mathrm{orig}} \otimes \boldsymbol{I}_2$, $\boldsymbol{B}_{\mathrm{orig}}$ the real-valued sparsity transform. $Q(\boldsymbol{u}|\boldsymbol{y})$ is fitted to $P(\boldsymbol{u}|\boldsymbol{y})$ by $\min_{\boldsymbol{\gamma} \succ \boldsymbol{0}} \phi$: a *convex* problem [24]. Used within an automatic decision architecture, convexity and robustness of inference become assets that are more important than smaller bias after a lot of human expert attention.

---

the repeat time (between phase encodes), several slices are acquired in an interleaved fashion, separated by slice gaps to avoid crosstalk [17].

Second, $\phi(\boldsymbol{\gamma})$ can be minimized very efficiently by a double loop algorithm [24]. The computationally intensive $\log|\boldsymbol{A}|$ term is concave in $\boldsymbol{\gamma}^{-1}$. Upper-bounding it tangentially by the affine $\boldsymbol{z}^T(\boldsymbol{\gamma}^{-1}) - g^*(\boldsymbol{z})$ at outer loop (OL) update points, the resulting $\phi_{\boldsymbol{z}} \geq \phi$ decouples and is minimized much more efficiently in inner loops (ILs). $\min_{\boldsymbol{\gamma} \succ \boldsymbol{0}} \phi_{\boldsymbol{z}}$ leaves us with

$$\min_{\boldsymbol{u}} \left\{ \phi_{\boldsymbol{z}}(\boldsymbol{u}) = \sigma^{-2} \|\boldsymbol{y} - \boldsymbol{X}\boldsymbol{u}\|^2 + 2\sum_j h_j^*(|s_j|) \right\}, \quad h_j^*(|s_j|) := \tau_j(z_j + (|s_j|/\sigma)^2)^{1/2}, \quad (2)$$

a penalized least squares problem. At convergence, $\boldsymbol{u}_* = \mathrm{E}_Q[\boldsymbol{u}|\boldsymbol{y}]$, $\gamma_j \leftarrow (z_j + |s_{*,j}/\sigma|^2)^{1/2}/\tau_j$. We can use iteratively reweighted least squares (IRLS), each step of which needs a linear system to be solved of the structure of $\boldsymbol{A}$. Refitting $\boldsymbol{z}$ (OL updates) is much harder: $\boldsymbol{z} \leftarrow (\boldsymbol{I} \otimes \boldsymbol{1}^T)\mathrm{diag}^{-1}(\boldsymbol{B}\boldsymbol{A}^{-1}\boldsymbol{B}^T) = (\boldsymbol{I} \otimes \boldsymbol{1}^T)(\sigma^{-2}\mathrm{Var}_Q[\boldsymbol{s}_j|\boldsymbol{y}])$. In terms of Gaussian (Markov) random fields, the inner optimization needs posterior *mean* computations only, while OL updates require bulk Gaussian *variances* [21, 15]. The reason why the double loop algorithm is much faster than previous approaches is that only few variance computations are required. The extension to complex-valued $\boldsymbol{u}$ is non-trivial only when it comes to IRLS search direction computations (see Appendix).

Given multi-slice data $(\boldsymbol{X}_t, \boldsymbol{y}_t)$, $t = 1, \ldots, T$, we can use an undirected hidden Markov model over image slices $\boldsymbol{u} = (\boldsymbol{u}_t) \in \mathbb{C}^{nT}$. By the stack-of-slices methodology, the likelihood potentials $P(\boldsymbol{y}_t|\boldsymbol{u}_t)$ are independent, and $P(\boldsymbol{u}_t)$ from above serves as single-node potential, based on $\boldsymbol{s}_t = \boldsymbol{B}\boldsymbol{u}_t$. If $\boldsymbol{s}_{t\rightarrow} := \boldsymbol{u}_t - \boldsymbol{u}_{t+1}$, the dependence between neighbouring slices is captured by additional Laplace coupling potentials $\prod_{i=1}^n e^{-\tau_{c,i}|(s_{t\rightarrow})_i/\sigma|}$. The variational parameters $\boldsymbol{\gamma}_t$ at each node are complemented by coupling parameters $\boldsymbol{\gamma}_{t\rightarrow} \in \mathbb{R}_+^n$. The Gaussian $Q(\boldsymbol{u}|\boldsymbol{y})$, $\boldsymbol{y} = (\boldsymbol{y}_t)$, has the same form as above with a huge $\boldsymbol{A} \in \mathbb{C}^{nT \times nT}$. Inheriting the Markov structure, it is a Gaussian linear dynamical system (LDS) with very high-dimensional states. How will an efficient extension of the double loop algorithm look like? The IL criterion $\phi_{\boldsymbol{z}}$ should be coupled between neighbouring slices, by way of potentials on $\boldsymbol{s}_{t\rightarrow}$. OL updates are more difficult to lift: we have to approximate marginal variances in a Gaussian LDS. We will do this by Kalman smoothing, approximating inversion in message computations (conversion from natural to moment parameters) by the Lanczos algorithm.

The central role of Gaussian covariance for approximating non-Gaussian posteriors has not been emphasized much in machine learning, where if Bayesian computations are intractable, simpler "variational Bayesian" concepts are routinely used, imposing factorization constraints on the posterior up front. While such constraints can be adjusted in light of the data, this is difficult and typically not done. Factorization assumptions are a double-edged sword: they radically simplify implementations, but result in non-convex algorithms, and half of the problem is left undone. Our approach offers an alternative: by using Lanczos on $Q(\boldsymbol{u}|\boldsymbol{y})$, we retain precisely the maximum-covariance directions of intermediate fits to the posterior, without running into combinatorial or non-convex problems. Finally, we place more varied sparsity penalties on the in-plane dimensions [24] than on the third one. This is justified by voxels typically being larger and spaced with a gap in the third dimension, with partial volume effects reducing sparsity. Moreover, a non-local sparsity transform along the third dimension would destroy the Markovian structure essential for efficient computation.

## 3 Approximate Inference over Multiple Slices

We aim to extend the single slice method of [24] to the hidden Markov extension, thereby reusing code whenever possible. The variational criterion is (1) with

$$h(\boldsymbol{\gamma}) = \sum_t h_t(\boldsymbol{\gamma}_t) + \mathrm{I}_{\{t<T\}} h_{t\rightarrow}(\boldsymbol{\gamma}_{t\rightarrow}), \ R = \sum_t R_t + \mathrm{I}_{\{t<T\}} R_{t\rightarrow}, \ \boldsymbol{\Gamma}_{t\rightarrow} := (\mathrm{diag}\,\boldsymbol{\gamma}_{t\rightarrow}) \otimes \boldsymbol{I}_2,$$
$$R_t = \sigma^{-2}\left(\|\boldsymbol{y}_t - \boldsymbol{X}_t\boldsymbol{u}_t\|^2 + \boldsymbol{s}_t^T\boldsymbol{\Gamma}_t^{-1}\boldsymbol{s}_t\right), \quad R_{t\rightarrow} = \sigma^{-2}\boldsymbol{s}_{t\rightarrow}^T\boldsymbol{\Gamma}_{t\rightarrow}^{-1}\boldsymbol{s}_{t\rightarrow}.$$

The coupling term $\log|\boldsymbol{A}|$ is upper-bounded ($\phi \leq \phi_{\boldsymbol{z}}$), so that the IL criterion $\phi_{\boldsymbol{z}}(\boldsymbol{u})$ is the sum of terms $\phi_{t,\boldsymbol{z}_t}(\boldsymbol{u}_t)$, $\phi_{t\rightarrow,\boldsymbol{z}_{t\rightarrow}}(\boldsymbol{s}_{t\rightarrow})$. Problems of the form $\min_{\boldsymbol{u}} \phi_{\boldsymbol{z}}$, jointly convex with couplings between neighbours, are routinely addressed in parallel convex optimization. In order to update $\boldsymbol{u}_t$, we consider its neighbours $\boldsymbol{u}_{t-1}$, $\boldsymbol{u}_{t+1}$ fixed, massaging $\phi_{t,\boldsymbol{z}_t}(\boldsymbol{u}_t) + \phi_{(t-1)\rightarrow,\boldsymbol{z}_{(t-1)\rightarrow}}(\boldsymbol{s}_{(t-1)\rightarrow}) + \phi_{t\rightarrow,\boldsymbol{z}_{t\rightarrow}}(\boldsymbol{s}_{t\rightarrow})$ into the form of [24]: $\tilde{\boldsymbol{B}} = (\boldsymbol{B}^T, \boldsymbol{I}, \boldsymbol{I})^T$, $\tilde{\boldsymbol{s}} = (\boldsymbol{s}_t^T, (\boldsymbol{u}_t - \boldsymbol{u}_{t-1})^T, (\boldsymbol{u}_t - \boldsymbol{u}_{t+1})^T)^T$, $\tilde{\boldsymbol{u}} = \boldsymbol{u}_t$. These updates can be run asynchronously in parallel, sending $\boldsymbol{u}_t$ to neighbours after every few IRLS steps.

For OL updates, we have to compute $\mathbf{z}_t = \sigma^{-2}(\mathbf{I} \otimes \mathbf{1}^T)\mathrm{Var}_Q[\mathbf{s}_t|\mathbf{y}]$ and $\mathbf{z}_{t\rightarrow} = \sigma^{-2}(\mathbf{I} \otimes \mathbf{1}^T)\mathrm{Var}_Q[\mathbf{s}_{t\rightarrow}|\mathbf{y}]$, where $Q(\mathbf{u}|\mathbf{y})$ is a Gaussian LDS (fixed $\boldsymbol{\gamma}$). To output a global criterion value, an estimate of $\log|\mathbf{A}|$ is required as well. We use the two-filter Kalman information smoother, which entails passing Gaussian-form messages along the chain in both directions. Once all messages are available, marginal (co)variances are computed at each node in parallel. Shift $Q(\mathbf{u}|\mathbf{y})$ to zero mean ($\mathrm{E}_Q[\mathbf{u}|\mathbf{y}] = \mathbf{u}_*$ is found in the IL). Denoting $N^U(\mathbf{A}) = N^U(\mathbf{u}|\mathbf{A}) := e^{-(1/2)\sigma^{-2}\mathbf{u}^T\mathbf{A}\mathbf{u}}$, $Q(\mathbf{u}|\mathbf{y})$ consists of single node potentials $\Phi_t(\mathbf{u}_t) = N^U(\mathbf{A}_t)$ and pair potentials $\Phi_{t\rightarrow}(\mathbf{s}_{t\rightarrow}) = N^U(\boldsymbol{\Gamma}_{t\rightarrow}^{-1})$, where $\mathbf{A}_t := \mathbf{X}_t^H\mathbf{X}_t + \mathbf{B}^T\boldsymbol{\Gamma}_t^{-1}\mathbf{B}$. Defining messages $M_{t\rightarrow}(\mathbf{u}_t) = N^U(\tilde{\mathbf{A}}_{t\rightarrow})$, $M_{\leftarrow t}(\mathbf{u}_t) = N^U(\tilde{\mathbf{A}}_{\leftarrow t})$, the usual message propagation equation is $M_{t\rightarrow}(\mathbf{u}_t) \propto \int M_{(t-1)\rightarrow}(\mathbf{u}_{t-1})\Phi_{(t-1)\rightarrow}(\mathbf{s}_{(t-1)\rightarrow})d\mathbf{u}_{t-1}\Phi_t(\mathbf{u}_t)$, so that

$$\tilde{\mathbf{A}}_{t\rightarrow} = \mathbf{A}_t + \mathcal{M}(\tilde{\mathbf{A}}_{(t-1)\rightarrow}, \boldsymbol{\Gamma}_{(t-1)\rightarrow}), \quad \mathcal{M}(\tilde{\mathbf{A}}, \boldsymbol{\Gamma}) := \boldsymbol{\Gamma}^{-1} - \boldsymbol{\Gamma}^{-1}(\tilde{\mathbf{A}} + \boldsymbol{\Gamma}^{-1})^{-1}\boldsymbol{\Gamma}^{-1}. \quad (3)$$

In the same way, $\tilde{\mathbf{A}}_{\leftarrow t} = \mathbf{A}_t + \mathcal{M}(\tilde{\mathbf{A}}_{\leftarrow(t+1)}, \boldsymbol{\Gamma}_{t\rightarrow})$. Denote $\mathcal{M}_{t\rightarrow} := \mathcal{M}(\tilde{\mathbf{A}}_{t\rightarrow}, \boldsymbol{\Gamma}_{t\rightarrow})$, $\mathcal{M}_{\leftarrow t} := \mathcal{M}(\tilde{\mathbf{A}}_{\leftarrow t}, \boldsymbol{\Gamma}_{(t-1)\rightarrow})$. Once all messages have been computed, the node marginal $Q(\mathbf{u}_t|\mathbf{y})$ has precision matrix $\tilde{\mathbf{A}}_t := \mathbf{A}_t + \mathcal{M}_{(t-1)\rightarrow} + \mathcal{M}_{\leftarrow(t+1)}$. If $\boldsymbol{\Psi} := (\boldsymbol{\delta}_1 - \boldsymbol{\delta}_2) \otimes \mathbf{I}$, the precision matrix of $Q(\mathbf{u}_t, \mathbf{u}_{t+1}|\mathbf{y})$ is $\mathrm{diag}(\tilde{\mathbf{A}}_{t\rightarrow}, \tilde{\mathbf{A}}_{\leftarrow(t+1)}) + \boldsymbol{\Psi}\boldsymbol{\Gamma}_{t\rightarrow}^{-1}\boldsymbol{\Psi}^T$, and $\mathbf{s}_{t\rightarrow} = \boldsymbol{\Psi}^T(\mathbf{u}_t^T, \mathbf{u}_{t+1}^T)^T$. $\mathrm{Cov}_Q[\mathbf{s}_{t\rightarrow}|\mathbf{y}]$ can be written in terms of $\tilde{\mathbf{A}}_{t+1}^{-1}$ and $\mathcal{M}_{t\rightarrow}$. Finally, by tracking normalization constants: $\log|\mathbf{A}| = \sum_{t<\tilde{t}}\log|\tilde{\mathbf{A}}_{t\rightarrow} + \boldsymbol{\Gamma}_{t\rightarrow}^{-1}| + \sum_{t>\tilde{t}}\log|\tilde{\mathbf{A}}_{\leftarrow t} + \boldsymbol{\Gamma}_{(t-1)\rightarrow}^{-1}| + \log|\tilde{\mathbf{A}}_{\tilde{t}}|$ for any $\tilde{t}$. In practice, we average over $\tilde{t}$. The algorithm is sketched in Algorithm 1.

---

**Algorithm 1** Double loop variational inference algorithm

---

**repeat**
  **if** first iteration **then**
    Default-initialize $\mathbf{z} \propto \mathbf{1}$, $\mathbf{u} = \mathbf{0}$.
  **else**
    Run Kalman smoothing to determine $\mathcal{M}_{t\rightarrow}$, and (in parallel) $\mathcal{M}_{\leftarrow t}$.
    Determine node variances $\mathbf{z}_t$, pair variances $\mathbf{z}_{t\rightarrow}$, and $\log|\mathbf{A}|$ from messages. Refit upper
    bound $\phi_{\mathbf{z}}$ to $\phi$ (tangent at $\boldsymbol{\gamma}$). Initialize $\mathbf{u} = \mathbf{u}_*$ (previous solution).
  **end if**
  **repeat**
    Distributed IRLS to minimize $\min_{\boldsymbol{\gamma}} \phi_{\mathbf{z}}$ w.r.t. $\mathbf{u}$.
    Each local update of $\mathbf{u}_t$ entails solving a linear system (conjugate gradients).
  **until** $\mathbf{u}_* = \mathrm{argmin}_{\mathbf{u}} \phi_{\mathbf{z}}$ converged
  Update $\gamma_j = (z_j + |s_{*,j}/\sigma|^2)^{1/2}/\tau_j$.
**until** outer loop converged

---

For reconstruction, we run parallel MAP estimation. Following [12], we smooth out the nondifferentiable $l_1$ penalty by $|s_j/\sigma| \approx (\varepsilon + |s_j/\sigma|^2)^{1/2}$ for very small $\varepsilon > 0$, then use nonlinear conjugate gradients with Armijo line search. Nodes return with $\nabla_{\mathbf{u}_t}\phi_{\mathbf{z}}$ at the line minimum $\mathbf{u}_t$, the next search direction is centrally determined and distributed (just a scalar has to be transferred). This is not the same as centralized CG: line searches are distributed and not done on the global criterion.

We briefly comment on how to approximate Kalman message passing by way of the Lanczos algorithm [8], full details are given in [22]. Gaussian (Markov) random field practitioners will appreciate the difficulties: there is no locally connected MRF structure, and the $Q(\mathbf{u}|\mathbf{y})$ are highly non-stationary, being fitted to a posterior with non-Gaussian statistics (edges in the image, *etc*). Message passing requires the inversion of a precision matrix $\mathbf{A}$. The idea behind Lanczos approximations is PCA: if $\mathbf{A} \approx \mathbf{U}\boldsymbol{\Lambda}\mathbf{U}^T$, $\boldsymbol{\Lambda}$ the $l \ll n$ smallest eigenvalues, $\mathbf{U}\boldsymbol{\Lambda}^{-1}\mathbf{U}^T$ is the PCA approximation of $\mathbf{A}^{-1}$. With matrices $\mathbf{A}$ of certain spectral decay, this representation can be approximated by Lanczos (see [24, 22] for details). For a low rank PCA approximation of $\tilde{\mathbf{A}}_{t\rightarrow}$, $\mathcal{M}_{t\rightarrow}$ has the same rank (see Appendix), which allows to run Gaussian message passing tractably. In a parallel implementation, the forward and backward filter passes run in parallel, passing low rank messages (the rank $k_m$ of these should be smaller than the rank $k_c$ for subsequent marginal covariance computations). On a lower level, both matrix-vector multiplications with $\mathbf{X}_t$ (FFT) and reorthogonalizations required during the Lanczos algorithm can easily be parallelized on commodity graphics hardware.

# 4 Sampling Optimization by Bayesian Experimental Design

With our multi-slice variational inference algorithm in place, we address sampling optimization by Bayesian sequential experimental design, following [24]. At slice $t$, the information gain score $\Delta(\boldsymbol{X}_*) := \log |\boldsymbol{I} + \boldsymbol{X}_* \mathrm{Cov}_Q[\boldsymbol{u}_t|\boldsymbol{y}] \boldsymbol{X}_*^T|$ is computed for a fixed number of phase encode candidates $\boldsymbol{X}_* \in \mathbb{C}^{d \times n}$ not yet in $\boldsymbol{X}_t$, the score maximizer is appended, and a novel measurement is acquired (for the maximizer only). $\Delta(\boldsymbol{X}_*)$ depends primarily on the marginal posterior *covariance* matrix $\mathrm{Cov}_Q[\boldsymbol{u}_t|\boldsymbol{y}]$, computed by Gaussian message passing just as variances in OL updates above (while a *single* value $\Delta(\boldsymbol{X}_*)$ can be estimated more efficiently, the dominating eigendirections of the global covariance matrix seem necessary to approximate *many* score values for different candidates $\boldsymbol{X}_*$). Once messages have been passed, scores can be computed in parallel at different nodes. A purely sequential approach, extending one design $\boldsymbol{X}_t$ by one encode in each round, is not tractable. In practice, we extend several node designs $\boldsymbol{X}_t$ in each round (a fixed subset $C_{\mathrm{it}} \subset \{1, \ldots, T\}$; "it" the round number). Typically, $C_{\mathrm{it}}$ repeats cyclically. This is approximate, since candidates are scored independently at each node. Certainly, $C_{\mathrm{it}}$ should not contain neighbouring nodes. In the interleaved stack-of-slices methodology, scan time is determined by the largest factor $\boldsymbol{X}_t$ (number of rows), so we strive for balanced designs here.

To sum up, our adaptive design optimization algorithm starts with an initial variational inference phase for a start-up design (low frequencies only), then runs through a fixed number of design rounds. Each round starts with Gaussian message passing, based on which scores are computed at nodes $t \in C_{\mathrm{it}}$, new measurements are acquired, and designs $\boldsymbol{X}_t$ are extended. Finally, variational inference is run for the extended model, using a small number of OL iterations (only one in our experiments). Time can be saved by basing the first OL update on the same messages and node marginal covariances than the design score computations (neglecting their change through new phase encodes).

# 5 Experiments

We present experimental results, comparing designs found by our Bayesian joint design optimization method against alternative choices on real MRI data. We use the model of Section 2, with the prior previously used in [24] (potentials of strength $\tau_a$ on wavelet coefficients, of strength $\tau_r$ on Cartesian finite differences). While the MRI signal $\boldsymbol{u}$ is complex-valued, phase contributions are mostly erroneous, and reconstruction as well as design optimization are improved by multiplying a further term $\prod_i e^{-(\tau_i/\sigma)|\Im(u_i)|}$ into each single node prior potential, easily incorporated into the generic setup by appending $\boldsymbol{I} \otimes \boldsymbol{\delta}_2^T$ to $\boldsymbol{B}$. We focus on *Cartesian* MRI (phase encodes are complete columns[3] in $k$-space): a more clinically relevant setting than spiral sampling treated in [24].

We use data of resolution $64 \times 64$ (in-plane) to test our approach with a serial implementation. While this is not a resolution of clinical relevance, a truly parallel implementation is required in order to run our method at resolutions $256 \times 256$ or beyond: an important point for future work.

## 5.1 Quality of Lanczos Variance Approximations

We begin with experiments to analyze the errors in Lanczos variance approximations. Recall from [24] that variances are underestimated. We work with a single slice of resolution $64 \times 64$, using a design $\boldsymbol{X}$ of 30 phase encodes, running a single common OL iteration (default-initialized $\boldsymbol{z}$), comparing different ways of continuing from there: exact $\boldsymbol{z}$ computations (Cholesky decomposition of $\boldsymbol{A}$) versus Lanczos approximations with different numbers of steps $k$. Results are in Figure 1.

While the relative approximation errors are rather large uniformly, there is a clear structure to them: the largest (and also the very smallest) true values $z_j$ are approximated significantly more accurately than smaller true values. This structure can be used to motivate why, in the presence of large errors over all coefficients, our inference still works well for sparse linear models, indeed in some cases *better* than if exact computations are used (Figure 1, upper right). The spectrum of $\boldsymbol{A}$ shows a roughly linear decay, so that the largest and smallest eigenvalues (and eigenvectors) are well-approximated

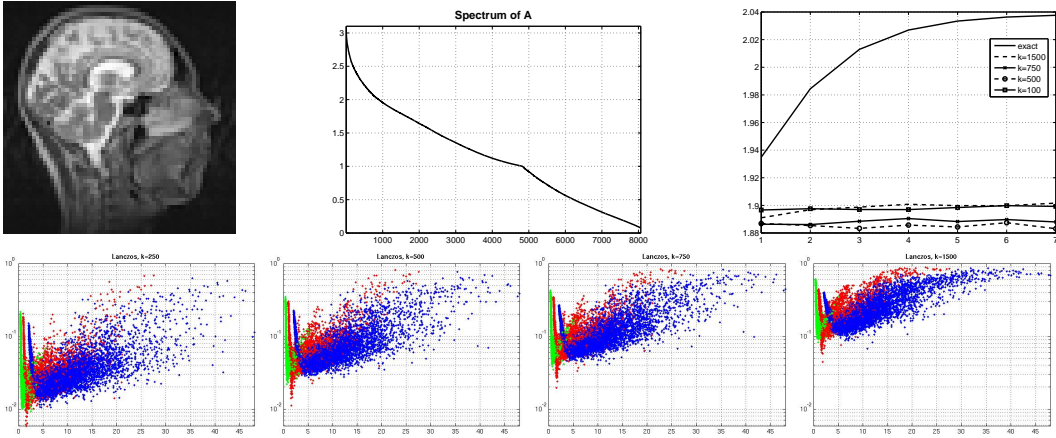

Figure 1: Lanczos approximations of Gaussian variances, at beginning of second OL iteration, $64 \times 64$ data (upper left). Spectral decay of inverse covariance matrix $\boldsymbol{A}$ roughly linear (upper middle). $l_2$ reconstruction error of posterior mean estimate after subsequent OL iterations, for exact variance computation vs. $k = 250, 500, 750, 1500$ Lanczos steps (upper right). Lower panel: Relative accuracy $z_j \mapsto z_{k,j}/z_j$ at beginning of second OL iteration, separately for "a" sites (on wavelet coefficients; red), "r" sites (on derivatives; blue), and "i" sites (on $\Im(\boldsymbol{u})$; green).

by Lanczos, while the middle part of the spectrum is not penetrated. Contributions to the largest values $z_j$ come dominatingly from small eigenvalues (large eigenvalues of $\boldsymbol{A}^{-1}$), explaining their smaller relative error. On the other hand, smaller values $z_j$ are strongly underestimated ($z_{k,j} \ll z_j$), which means that the selective shrinkage effect underlying sparse linear models (shrink most coefficients strongly, but some not at all) is *strengthened* by these systematic errors. Finally, the IL penalties are $\tau_j(z_j + |s_j/\sigma|^2)^{1/2}$, enforcing sparsity more strongly for smaller $z_j$. Therefore, Lanczos approximation errors lead to strengthened sparsity in subsequent ILs, but least so for sites with largest true $z_j$. As an educated guess, this effect might even compensate for the fact that Laplace potentials may not be sparse enough for natural images.

## 5.2  Joint Design Optimization

We use sagittal head scan data of resolution $64 \times 64$ in-plane, 32 slices, acquired on a Siemens 3T scanner (phase direction anterior-posterior), see [22] for further details. We consider joint and independent MAP reconstruction (for the latter, we run nonlinear CG separately for each slice), for a number of different design choices: $\{\boldsymbol{X}_t\}$ optimized jointly by our method here [op-jt]; each $\boldsymbol{X}_t$ optimized separately, by running the complex variant of [24] on slice $\boldsymbol{u}_t$ [op-sp]; $\boldsymbol{X}_t = \boldsymbol{X}$ for all $t$, with $\boldsymbol{X}$ optimized on the most detailed slice (number 16, Figure 2, row 2 middle) [op-eq]; and encodes of each $\boldsymbol{X}_t$ drawn at random, from the density proposed in [12] [rd], respecting the typical spectral decay of images [4] (all designs contain the 8 lowest-frequency encodes). Results for rd are averaged over ten repetitions. For all setups but op-eq, $\boldsymbol{X}_t$ are different across $t$. Hyperparameters are adjusted based on MAP reconstruction results for a fixed design picked ad hoc ($\tau_a = \tau_r = 0.01$, $\tau_i = 0.1$ in-plane; $\tau_c = 0.08$ between slices), then used for all design optimization and MAP reconstruction runs. We run the op-jt optimization with an odd-even schedule $\{C_{\text{it}}\}$ (all odd (even) $t \in 0, \dots, T-1$ for odd (even) "it"); results for two other schedules of period four come out very similar, but require more running time. For variational inference, we run 6 OL iterations in the initial, 1 OL iteration in each design round, with up to 30 IL steps (ILs in design rounds typically converged in 2–3 steps). The rank parameters (number of Lanczos steps)[4] were $k_m = 100$, $k_c = 250$ (here, $\boldsymbol{u}_t$ has $\tilde{n} = 8192$ real coefficients). Results are given in Figure 2.

First, across all designs, joint MAP reconstruction improves significantly upon independent MAP reconstruction. This improvement is strongest by far for op-jt (see Figure 2, rows 3,4), which for joint reconstruction improves on all other variants significantly, especially with 16–30 phase

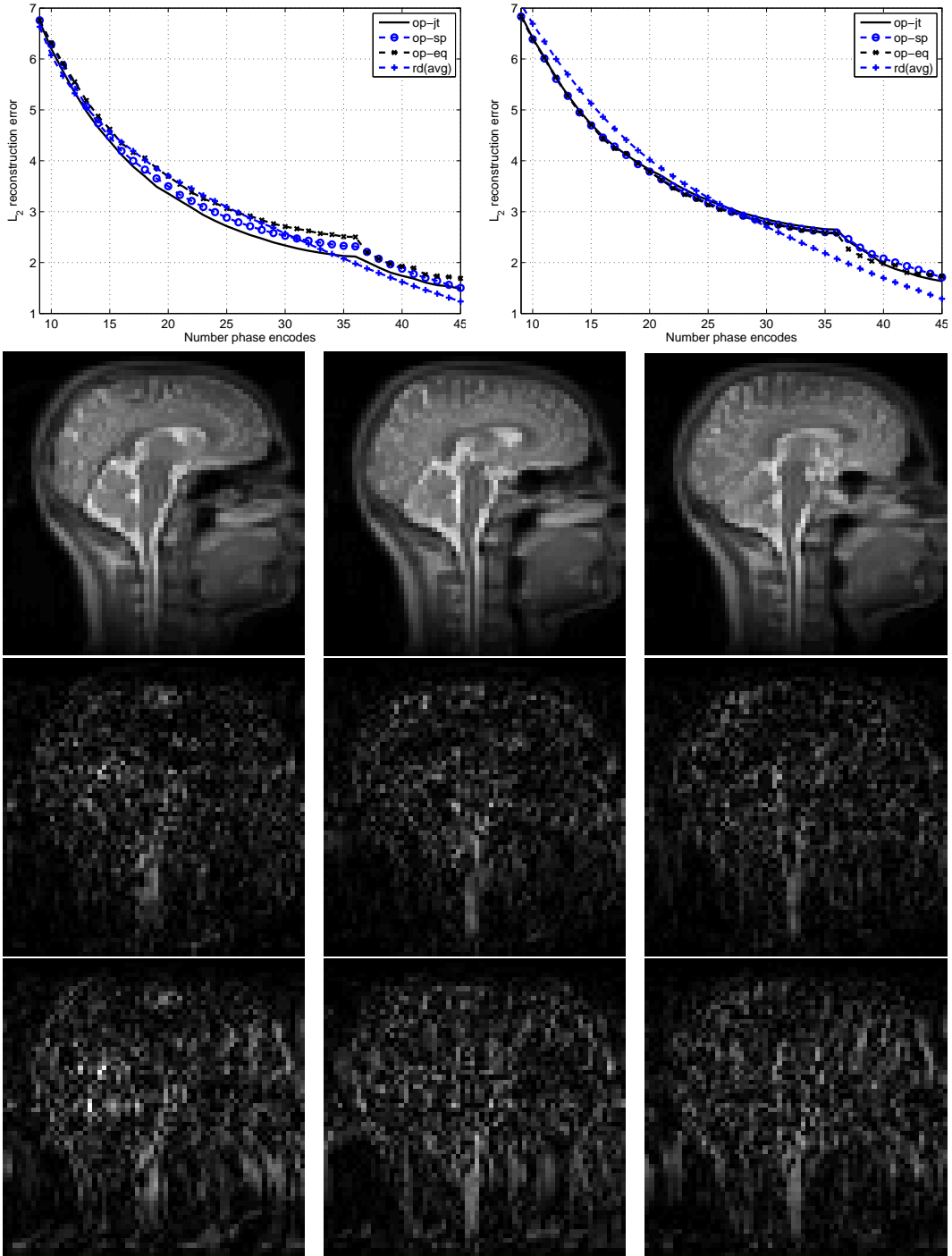

Figure 2: Top row: $l_2$ reconstruction errors $\||\hat{\boldsymbol{u}}_{\mathrm{MAP}}| - |\boldsymbol{u}_{\mathrm{true}}|\|$ of MAP reconstruction for different measurement designs. Left: joint MAP reconstruction; right: independent MAP reconstruction of each slice. op-jt: $\{\boldsymbol{X}_t\}$ optimized jointly; op-sp: $\boldsymbol{X}_t$ optimized separately for each slice; op-eq: $\boldsymbol{X}_t = \boldsymbol{X}$, optimized on slice 16; rd: $\boldsymbol{X}_t$ variable density drawn at random (averaged over 10 repetitions).
Rows 2–4: Images for op-jt (25 encodes), slices 15–17. Row 2: true images (range 0–0.35). Row 3: errors joint MAP. Row 4: errors indep. MAP (range 0–0.08).

encodes, where scan time is reduced by a factor 2–4 (Nyquist sampling requires 64 phase encodes). op-eq does worst in this domain: with a model of dependencies between slices in place, it pays

off to choose different $\boldsymbol{X}_t$ for each slice. `rnd` does best from about 35 phase encodes on. While this suboptimal behaviour of our optimization will be analyzed more closely in future work, it is our experience so far that the gain in using greedy sequential Bayesian design optimization over simpler choices is generally largest below 1/2 Nyquist.

## 6 Conclusions

We showed how to implement MRI sampling optimization by Bayesian sequential experimental design, jointly over a stack of neighbouring slices, extending the single slice technique of [24]. Restricting ourselves to undersampling of Cartesian encodes, our method can be applied in practice whenever dense Cartesian sampling is well under control (sequence modification is limited to skipping encodes). We exploit the hidden Markov structure of the model by way of a Lanczos approximation of Kalman smoothing. While the latter has been proposed for spatial statistics applications [20, 25], it has not been used for non-Gaussian approximate inference before, nor in the context of sparsity-favouring image models or non-linear experimental design. Our method is a general alternative to structured variational mean field approximations typically used for non-Gaussian dynamical systems, in that dominating covariances are tracked *a posteriori*, rather than eliminating most of them *a priori* through factorization assumptions. In a first study, we obtain encouraging results in the range below 1/2 Nyquist. In future work, we will develop a truly parallel implementation, with which higher resolutions can be processed. We are considering extensions of our design optimization technology to 3D MRI[5] and to parallel MRI with receiver coil arrays [19, 9], whose combination with $k$-space undersampling can be substantially more powerful than each acceleration technique on its own [13].

### Appendix

For norm potentials, $h_j^*(\boldsymbol{s}_j) = h_j^*(\|\boldsymbol{s}_j\|)$, and the Hessians to solve for IRLS Newton directions do not have the form of $\boldsymbol{A}$ anymore. In order to understand this, note that we do not use complex calculus here: $s \mapsto |s|$ is not complex differentiable at any $s \in \mathbb{C}$. Rather, we use the $\mathbb{C} \to \mathbb{R}^2$ embedding, then standard real-valued optimization for variables twice the size. If $\theta_j := (h_j^*)'$, $\rho_j := (h_j^*)''$ at $\|\boldsymbol{s}_j\| \neq 0$, then using $\nabla_{\boldsymbol{s}_j} \|\boldsymbol{s}_j\| = \boldsymbol{s}_j / \|\boldsymbol{s}_j\|$, we have $\nabla\nabla_{\boldsymbol{s}_j} h_j^* = \rho_j \boldsymbol{I}_2 + \kappa_j^2 (\|\boldsymbol{s}_j\|^2 \boldsymbol{I}_2 - \boldsymbol{s}_j \boldsymbol{s}_j^T)$, $\kappa_j := (\theta_j/\|\boldsymbol{s}_j\| - \rho_j)^{1/2}/\|\boldsymbol{s}_j\|$. Since $\|\boldsymbol{s}_j\|^2 \boldsymbol{I}_2 - \boldsymbol{s}_j \boldsymbol{s}_j^T = \boldsymbol{\nu} \boldsymbol{s}_j (\boldsymbol{\nu} \boldsymbol{s}_j)^T$, $\boldsymbol{\nu} := \boldsymbol{\delta}_2 \boldsymbol{\delta}_1^T - \boldsymbol{\delta}_1 \boldsymbol{\delta}_2^T$, the Hessian is $\boldsymbol{X}^H \boldsymbol{X} + \boldsymbol{B} \boldsymbol{H}^{(s)} \boldsymbol{B}^T$. If $\hat{\boldsymbol{s}} := ((\text{diag } \boldsymbol{\kappa}) \otimes \boldsymbol{\nu}) \boldsymbol{s}$, then for any $\boldsymbol{v} \in \mathbb{R}^{2q}$: $\boldsymbol{H}^{(s)} \boldsymbol{v} = ((\text{diag } \boldsymbol{\rho}) \otimes \boldsymbol{I}_2) \boldsymbol{v} + ((\text{diag } \boldsymbol{w}) \otimes \boldsymbol{I}_2) \hat{\boldsymbol{s}}$, where $w_j := \boldsymbol{v}_j^T \hat{\boldsymbol{s}}_j$, $j = 1, \dots, q$, which shows how to compute Hessian matrix-vector multiplications, thus to implement IRLS steps in the complex-valued case.

Recall that messages are passed, alternating between $\tilde{\boldsymbol{A}}_{t\to}$ and $\mathcal{M}_{t\to}$ matrices. For a PCA approximation $\tilde{\boldsymbol{A}}_{t\to} \approx \boldsymbol{Q}_{t\to} \boldsymbol{T}_{t\to} \boldsymbol{Q}_{t\to}^T$, $\boldsymbol{Q}_{t\to} \in \mathbb{R}^{\tilde{n} \times k_m}$ orthonormal, $\boldsymbol{T}_{t\to}$ tridiagonal (obtained by running $k_m$ Lanczos steps for $\tilde{\boldsymbol{A}}_{t\to}$), low rank algebra gives

$$\mathcal{M}_{t\to} = \mathcal{M}(\tilde{\boldsymbol{A}}_{t\to}, \boldsymbol{\Gamma}_{t\to}^{-1}) = \boldsymbol{Q}_{t\to} \left( \boldsymbol{T}_{t\to}^{-1} + \boldsymbol{Q}_{t\to}^T \boldsymbol{\Gamma}_{t\to} \boldsymbol{Q}_{t\to} \right)^{-1} \boldsymbol{Q}_{t\to}^T = \boldsymbol{V}_{t\to} \boldsymbol{V}_{t\to}^T, \quad \boldsymbol{V}_{t\to} \in \mathbb{R}^{\tilde{n} \times k_m},$$

computed in $O(n\, k_m^2)$ by way of a Cholesky decomposition. Now, $\tilde{\boldsymbol{A}}_{(t+1)\to} = \boldsymbol{A}_{t+1} + \boldsymbol{V}_{t\to} \boldsymbol{V}_{t\to}^T$ becomes the precision matrix for the next Lanczos run: MVMs have additional complexity of $O(n\, k_m)$. Given all messages, node covariances are PCA-approximated by running Lanczos on $\boldsymbol{A}_t + \boldsymbol{V}_{(t-1)\to} \boldsymbol{V}_{(t-1)\to}^T + \boldsymbol{V}_{\leftarrow(t+1)} \boldsymbol{V}_{\leftarrow(t+1)}^T$ for $k_c$ iterations. Pair variances $\text{Var}_Q[\boldsymbol{s}_{t\to}|\boldsymbol{y}]$ are estimated by running Lanczos on vectors of size $2\tilde{n}$ (say for $k_c/2$ iterations; the precision matrix is given in Section 3). More details are given in [22].

### Acknowledgments

This work is partly funded by the Excellence Initiative of the German research foundation (DFG). It is part of an ongoing collaboration with Rolf Pohmann, Hannes Nickisch and Bernhard Schölkopf, MPI for Biological Cybernetics, Tübingen, where data for this study has been acquired.

## Footnotes

[1]While parallel MRI is becoming the standard, its use is not straightforward. The sensitivity maps are unknown up front, depend partly on what is scanned, and their reliable estimation can be difficult.

[2]"Stack-of-slices" acquisition along the z axis works by transmitting a narrow-band excitation pulse while applying a magnetic field gradient linear in z. If the echo time (between excitation and readout) is shorter than

[3]Our data are sagittal head scans, where the frequency encode direction (along which oversampling is possible at no extra cost) is typically chosen vertically (the longer anatomic axis).

[4]We repeated op-jt partly with $k_m = 250$, with very similar MAP reconstruction errors for the final designs, but significantly longer run time.

[5]In 3D MRI, image volumes are acquired without slice selection, using phase encoding along two dimensions. There are no unmeasured slice gaps and voxels are isotropic, but scan time is much longer.

# References

[1] M.A. Bernstein, K.F. King, and X.J. Zhou. *Handbook of MRI Pulse Sequences*. Elsevier Academic Press, 1st edition, 2004.

[2] R. Bracewell. *The Fourier Transform and Its Applications*. McGraw-Hill, 3rd edition, 1999.

[3] E. Candès, J. Romberg, and T. Tao. Robust uncertainty principles: Exact signal reconstruction from highly incomplete frequency information. *IEEE Trans. Inf. Theo.*, 52(2):489–509, 2006.

[4] H. Chang, Y. Weiss, and W. Freeman. Informative sensing. Technical Report 0901.4275v1 [cs.IT], ArXiv, 2009.

[5] D. Donoho. Compressed sensing. *IEEE Trans. Inf. Theo.*, 52(4):1289–1306, 2006.

[6] A. Garroway, P. Grannell, and P. Mansfield. Image formation in NMR by a selective irradiative pulse. *J. Phys. C: Solid State Phys.*, 7:L457–L462, 1974.

[7] M. Girolami. A variational method for learning sparse and overcomplete representations. *N. Comp.*, 13:2517–2532, 2001.

[8] G. Golub and C. Van Loan. *Matrix Computations*. Johns Hopkins University Press, 3rd edition, 1996.

[9] M. A. Griswold, P. M. Jakob, R. M. Heidemann, M. Nittka, V. Jellus, J. Wang, B. Kiefer, and A. Haase. Generalized autocalibrating partially parallel acquisitions (GRAPPA). *Magn. Reson. Med.*, 47(6):1202–10, 2002.

[10] P. Lauterbur. Image formation by induced local interactions: Examples employing nuclear magnetic resonance. *Nature*, 242:190–191, 1973.

[11] A. Levin, W. Freeman, and F. Durand. Understanding camera trade-offs through a Bayesian analysis of light field projections. In *European Conference on Computer Vision*, LNCS 5305, pages 88–101. Springer, 2008.

[12] M. Lustig, D. Donoho, and J. Pauly. Sparse MRI: The application of compressed sensing for rapid MR imaging. *Magn. Reson. Med.*, 85(6):1182–1195, 2007.

[13] M. Lustig and J. Pauly. SPIR-iT: Iterative self consistent parallel imaging reconstruction from arbitrary k-space. *Magn. Reson. Med.*, 2009. In print.

[14] B. Madore, G. Glover, and N. Pelc. Unalising by Fourier-encoding the overlaps using the temporal dimension (UNFOLD), applied to cardiac imaging and fMRI. *Magn. Reson. Med.*, 42:813–828, 1999.

[15] D. Malioutov, J. Johnson, and A. Willsky. Low-rank variance estimation in large-scale GMRF models. In *ICASSP*, 2006.

[16] G. Marseille, R. de Beer, M. Fuderer, A. Mehlkopf, and D. van Ormondt. Nonuniform phase-encode distributions for MRI scan time reduction. *J. Magn. Reson. B*, 111(1):70–75, 1996.

[17] D. McRobbie, E. Moore, M. Graves, and M. Prince. *MRI: From Picture to Proton*. Cambridge University Press, 2nd edition, 2007.

[18] C. Mistretta, O. Wieben, J. Velikina, W. Block, J. Perry, Y. Wu, K. Johnson, and Y. Wu. Highly constrained backprojection for time-resolved MRI. *Magn. Reson. Med.*, 55:30–40, 2006.

[19] K. Pruessmann, M. Weiger, M. Scheidegger, and P. Boesiger. SENSE: Sensitivity encoding for fast MRI. *Magn. Reson. Med.*, 42:952–962, 1999.

[20] M. Schneider and A. Willsky. Krylov subspace algorithms for space-time oceanography data assimilation. In *IEEE International Geoscience and Remote Sensing Symposium*, 2000.

[21] M. Schneider and A. Willsky. Krylov subspace estimation. *SIAM J. Comp.*, 22(5):1840–1864, 2001.

[22] M. Seeger. Speeding up magnetic resonance image acquisition by Bayesian multi-slice adaptive compressed sensing. Supplemental Appendix, 2010.

[23] M. Seeger and H. Nickisch. Compressed sensing and Bayesian experimental design. In *ICML 25*, 2008.

[24] M. Seeger, H. Nickisch, R. Pohmann, and B. Schölkopf. Bayesian experimental design of magnetic resonance imaging sequences. In *NIPS 21*, pages 1441–1448, 2009.

[25] D. Treebushny and H. Madsen. On the construction of a reduced rank square-root Kalman filter for efficient uncertainty propagation. *Future Gener. Comput. Syst.*, 21(7):1047–1055, 2005.

[26] J. Tsao, P. Boesinger, and K. Pruessmann. k-t BLAST and k-t SENSE: Dynamic MRI with high frame rate exploting spatiotemporal correlations. *Magn. Reson. Med.*, 50:1031–1042, 2003.

[27] F. Wajer. *Non-Cartesian MRI Scan Time Reduction through Sparse Sampling*. PhD thesis, Delft University of Technology, 2001.

[28] J. Weaver, Y. Xu, D. Healy, and L. Cromwell. Filtering noise from images with wavelet transforms. *Magn. Reson. Med.*, 21(2):288–295, 1991.

